# Generalized Model Selection For Unsupervised Learning In High Dimensions

**Shivakumar Vaithyanathan**
IBM Almaden Research Center
650 Harry Road
San Jose, CA 95136
Shiv@almaden.ibm.com

**Byron Dom**
IBM Almaden Research Center
650 Harry Road
San Jose, CA 95136
dom@almaden.ibm.com

## Abstract

We describe a Bayesian approach to model selection in unsupervised learning that determines both the feature set and the number of clusters. We then evaluate this scheme (based on marginal likelihood) and one based on cross-validated likelihood. For the Bayesian scheme we derive a closed-form solution of the marginal likelihood by assuming appropriate forms of the likelihood function and prior. Extensive experiments compare these approaches and all results are verified by comparison against ground truth. In these experiments the Bayesian scheme using our objective function gave better results than cross-validation.

## 1 Introduction

Recent efforts define the model selection problem as one of estimating the number of clusters[10, 17]. It is easy to see, particularly in applications with large number of features, that various choices of feature subsets will reveal different structures underlying the data. It is our contention that this interplay between the feature subset and the number of clusters is essential to provide appropriate views of the data. We thus define the problem of model selection in clustering as selecting both the number of clusters *and* the feature subset. Towards this end we propose a unified objective function whose arguments include the both the feature space and number of clusters. We then describe two approaches to model selection using this objective function. The first approach is based on a Bayesian scheme using the marginal likelihood for model selection. The second approach is based on a scheme using cross-validated likelihood. In section 3 we apply these approaches to document clustering by making assumptions about the document generation model. Further, for the Bayesian approach we derive a closed-form solution for the marginal likelihood using this document generation model. We also describe a heuristic for initial feature selection based on the distributional clustering of terms. Section 5 describes the experiments and our approach to validate the proposed models and algorithms. Section 6 reports and discusses the results of our experiments and finally section 7 provides directions for future work.

# 2 Model selection in clustering

Model selection approaches in clustering have primarily concentrated on determining the number of components/clusters. These attempts include Bayesian approaches [7,10], MDL approaches [15] and cross-validation techniques [17]. As noticed in [17] however, the optimal number of clusters is dependent on the feature space in which the clustering is performed. Related work has been described in [7].

## 2.1 A generalized model for clustering

Let $D$ be a data-set consisting of "patterns" $\{d_1, .., d_v\}$, which we assume to be represented in some feature space $T$ with dimension $M$. The particular problem we address is that of clustering $D$ into groups such that its likelihood described by a probability model $P(D^T|\Omega)$, is maximized, where $D^T$ indicates the representation of $D$ in feature space $T$ and $\Omega$ is the structure of the model, which consists of the number of clusters, the partitioning of the feature set (explained below) and the assignment of patterns to clusters. This model is a weighted sum of models $\{P(D^T|\Omega, \xi)|\xi \in \mathbb{R}^m\}$ where $\xi$ is the set of all parameters associated with $\Omega$. To define our model we begin by assuming that the feature space $T$ consists of two sets: $U$ - useful features and $N$ - noise features. Our feature-selection problem will thus consist of partitioning $T$ (into $U$ and $N$ ) for a given number of clusters.

*Assumption 1 The feature sets represented by $U$ and $N$ are conditionally independent*

$$P(D^T|\Omega, \xi) = P(D^N \mid \Omega, \xi)\, P(D^U \mid \Omega, \xi) \qquad (1)$$

*where $D^N$ indicates data represented in the noise feature space and $D^U$ indicates data represented in useful feature space.*

Using assumption 1 and assuming that the data is independently drawn, we can rewrite equation (1) as

$$P(D^T|\Omega, \xi) = \left\{ \prod_{i=1}^{v} p(d_i^N \mid \xi^N) \cdot \prod_{k=1}^{K} \prod_{j \in D_k} p(d_j^U \mid \xi_k^U) \right\} \qquad (2)$$

where $v$ is the number of patterns in $D$, $p(d_i^U \mid \xi^U)$ is the probability of $d_i^U$ given the parameter vector $\xi_v^U$ and $p(d_i^N \mid \xi^N)$ is the probability of $d_i^N$ given the parameter vector $\xi^N$. Note that while the explicit dependence on $\Omega$ has been removed in this notation, it is implicit in the number of clusters $K$ and the partition of $T$ into $N$ and $U$.

## 2.2 Bayesian approach to model selection

The objective function, represented in equation (2) is not regularized and attempts to optimize it directly may result in the set $N$ becoming empty - resulting in overfitting. To overcome this problem we use the *marginal likelihood*[2].

*Assumption 2 All parameter vectors are independent.* $\quad \pi(\xi) = \pi(\xi^N) \cdot \displaystyle\prod_{k=1}^{K} \pi(\xi_k^U)$

where the $\pi(...)$ denotes a Bayesian *prior* distribution. The marginal likelihood, using assumption 2, can be written as

$$P(D^T \mid \Omega) = \int_{\Xi^N} \left[ \prod_{i=1}^{v} p(d_i^N \mid \xi^N) \right] \pi(\xi^N)\, d\xi^N \cdot \prod_{k=1}^{K} \int_{\Xi^U} \left[ \prod_{i \in D_k} p(d_i^U \mid \xi_k^U) \right] \pi(\xi_k^U)\, d\xi_k^U \quad (3)$$

where $\Xi^N, \Xi^U$ are integral limits appropriate to the particular parameter spaces. These will be omitted to simplify the notation.

## 3.0 Document clustering

Document clustering algorithms typically start by representing the document as a "bag-of-words" in which the features can number $\sim 10^4$ to $10^5$. Ad-hoc dimensionality reduction techniques such as stop-word removal, frequency based truncations [16] and techniques such as LSI [5] are available. Once the dimensionality has been reduced, the documents are usually clustered into an *arbitrary* number of clusters .

### 3.1 Multinomial models

Several models of text generation have been studied[3]. Our choice is *multinomial* models using term counts as the features. This choice introduces another parameter indicating the probability of the $N$ and $U$ split. This is equivalent to assuming a generation model where for each document the number of noise and useful terms are determined by a probability $\theta^S$ and then the terms in a document are "drawn" with a probability ($\theta^n$ or $\theta_k^u$ ).

### 3.2 Marginal likelihood / stochastic complexity

To apply our Bayesian objective function we begin by substituting multinomial models into (3) and simplifying to obtain

$$P(D \mid \Omega) = \binom{t^N + t^U}{t^N} \int [(\theta^S)^{t^N}(1 - \theta^S)^{t^U}]\pi(\theta^S)d\theta^S \cdot$$

$$\left[\prod_{k=1}^{K} \prod_{i \in D_k} \binom{t_i^U}{\{t_{i,u} \mid u \in U\}}\right] \int \left[\prod_{u \in U}(\theta_k^u)^{t_{i,u}}\right]\pi(\theta_k^U)d\theta_k^U \cdot \quad (4)$$

$$\left[\prod_{j=1}^{v} \binom{t_j^N}{\{t_{j,n} \mid n \in N\}}\right] \int \left[\prod_{n \in N}(\theta^n)^{t_{j,n}}\right]\pi(\theta^N)d\theta^N$$

where $\binom{\cdots}{\{\ldots\}}$ is the multinomial coefficient, $t_{i,u}$ is the number of occurrences of the feature term $u$ in document $i$, $t_i^U$ is the total number of all *useful* features (terms) in document $i$ ($t_i^U = \sum_u t_{i,u}$, $t_i^N$, and $t_{i,n}$ are to be interpreted similar to above but for noise features , $\binom{n}{k} = \frac{n!}{k!(n-k)!}$ , $t^N$ is the total number of all *noise* features in all patterns and $t^U$ is the total number of all useful features in all patterns.

To solve (4) we still need a form for the *priors* $\{\pi(\ldots)\}$. The Beta family is *conjugate* to the Binomial family [2] and we choose the Dirichlet distribution (multiple Beta) as the form for both $\pi(\theta_k^U)$ and $\pi(\theta^N)$ and the Beta distribution for $\pi(\theta^S)$. Substituting these into equation (8) and simplifying yields

$$P(D \mid \Omega) = \left[\frac{\Gamma(\gamma_a + \gamma_b)}{\Gamma(\gamma_a)\Gamma(\gamma_b)}\frac{\Gamma(t^N + \gamma_a)\Gamma(t^U + \gamma_b)}{\Gamma(t^U + t^N + \gamma_a + \gamma_b)}\right] \cdot \left[\frac{\Gamma(\beta)}{\Gamma(\beta + t^N)}\prod_{n \in N}\frac{\Gamma(\beta_n + t^n)}{\Gamma(\beta_n)}\right]$$

$$\left[\frac{\Gamma(\sigma)}{\Gamma(\sigma + v)}\prod_{k=1}^{K}\frac{\Gamma(\sigma_k + |D_k|)}{\Gamma(|D_k|)}\right] \cdot \left[\prod_{k=1}^{K}\frac{\Gamma(a)}{\Gamma(a + t^{U(k)})}\prod_{u \in U}\frac{\Gamma(a_u + t_k^u)}{\Gamma(a_u)}\right]$$

$$(5)$$

where $\beta_i$ and $a_u$ are the hyper-parameters of the Dirichlet prior for noise and useful features respectively, $\beta = \sum_{n \in N} \beta_n$ , $a = \sum_{u \in U} a_u$, $\sigma = \sum_{k=1} \sigma_k$ and $\Gamma()$ is the "gamma" function. Further, $\gamma_a, \gamma_b$ are the hyper parameters of the Beta prior for the split probability, $|D_k|$ is the number of documents in cluster k and $_t U(k$ is computed as $\sum_{i \in D_k} t_i^l$. The results reported for our evaluation will be the negative of the log of equation (5), which (following Rissanen [14]) we refer to as Stochastic Complexity (*SC*). In our experiments all values of the hyper-parameters $\beta_i, a_i$ $\sigma_k, \gamma_a$ and $\gamma_b$ are set equal to 1 yielding uniform priors.

## 3.3 Cross-Validated likelihood

To compute the cross validated likelihood using multinomial models we first substitute the multinomial functional forms, using the MLE found using the training set. This results in the following equation

$$P(CV^T \mid \Omega^p) = \left\{ [(\widetilde{\theta^S})^{t_{cv}^N} \cdot (1 - \widetilde{\theta^S})^{t_{cv}^U}] \prod_{i=1}^{v_{test}} p(cv_i^N \mid \widetilde{\theta^N}) \cdot \prod_{k=1}^{K} \prod_{j \in D_k} p(cv_j^U \mid \widetilde{\theta_{k(i)}^U}) \cdot p(c_k) \right. (6)$$

where $\widetilde{\theta^S}$, $\widetilde{\theta^N}$ and $\widetilde{\theta_{k(i)}^U}$ are the MLE of the appropriate parameter vectors. For our implementation of *MCCV*, following the suggestion in [17], we have used a 50% split of the training and test set. For the *vCV* criterion although a value of v = 10 was suggested therein, for computational reasons we have used a value of v = 5.

## 3.4 Feature subset selection algorithm for document clustering

As noted in section 2.1, for a feature-set of size $M$ there are a total of $2^M$ partitions and for large $M$ it would be computationally intractable to search through all possible partitions to find the optimal subset. In this section we propose a heuristic method to obtain a subset of tokens that are topical (indicative of underlying topics) and can be used as features in the bag-of-words model to cluster documents.

### 3.4.1 Distributional clustering for feature subset selection

Identifying content-bearing and topical terms, is an active research area [9]. We are less concerned with modeling the exact distributions of individual terms as we are with simply identifying groups of terms that are topical. Distributional clustering (DC), apparently first proposed by Pereira et al [13], has been used for feature selection in supervised text classification [1] and clustering images in video sequences [9]. We hypothesize that function, content-bearing and topical terms have different distributions over the documents. DC helps reduce the size of the search space for feature selection from $2^M$ to $2^C$, where $C$ is the number of clusters produced by the DC algorithm. Following the suggestions in [9], we compute the following histogram for each token. The first bin consists of the number of documents with zero occurrences of the token, the second bin is the number of documents consisting of a single occurrence of the token and the third bin is the number of documents that contain more two or more occurrences of the term. The histograms are clustered using *relative entropy* $\Delta(. \parallel .)$ as

a distance measure. For two terms with probability distributions $p_1(.)$ and $p_2(.)$, this is given by [4]:

$$\Delta(p_1(t) \parallel p_2(t)) \equiv \sum_t p_1(t) \log \frac{p_1(t)}{p_2(t)} \tag{7}$$

We use a k-means-style algorithm in which the histograms are normalized to sum to one and the sum in equation (7) is taken over the three bins corresponding to counts of 0,1, and $\geq 2$. During the assignment-to-clusters step of k-means we compute $\Delta(p_w \parallel p_{c_k})$ (where $p_w$ is the normalized histogram for term $w$ and $p_{c_k}(t)$ is the centroid of cluster $k$) and the term $w$ is assigned to the cluster for which this is minimum [13,8].

## 4.0 Experimental setup

Our evaluation experiments compared the clustering results against human-labeled ground truth. The corpus used was the AP Reuters Newswire articles from the TREC-6 collection. A total of 8235 documents, from the routing track, existing in 25 classes were analyzed in our experiments. To simplify matters we disregarded multiple assignments and retained each document as a member of a single class.

### 4.1 Mutual information as an evaluation measure of clustering

We verify our models by comparing our clustering results against pre-classified text. We force all clustering algorithms to produce exactly as many clusters as there are classes in the pre-classified text and we report the mutual information[4] (MI) between the cluster labels and pre-classified class labels

## 5.0 Results and discussions

After tokenizing the documents and discarding terms that appeared in less than 3 documents we were left with 32450 unique terms. We experimented with several numbers of clusters for DC but report only the best (lowest SC) for lack of space. For each of these clusters we chose the best of 20 runs corresponding to different random starting clusters. Each of these sets includes one cluster that consists of high-frequency words and upon examination were found to contain primarily *function* words, which we eliminated from further consideration. The remaining non-function-word clusters were used as feature sets for the clustering algorithm. Only combinations of feature sets that produced good results were used for further document clustering runs.

We initialized the EM algorithm using k-means algorithm - other initialization schemes are discussed in [11]. The feature vectors used in this k-means initialization were generated using the pivoted normal weighting suggested in [16]. All parameter vectors $\theta_k^U$ and $\theta^N$ were estimated using Laplace's Rule of Succession[2]. Table 1 shows the best results of the *SC* criterion, the *vCV* and *MCCV* using the feature subsets selected by the different combinations of distributional clusters. The feature subsets are coded as FSXP where X indicates the number of clusters in the distributional clustering and P indicates the cluster number(s) used as *U*. For *SC* and MI all results reported are averages over 3 runs of the k-means+EM combination with different initialization fo k-means. For clarity, the MI numbers reported are normalized such that the theoretical maximum is 1.0. We also show comparisons against no feature selection (NF) and LSI.

For LSI, the principal 165 eigenvectors were retained and k-means clustering was performed in the reduced dimensional space. While determining the number of clusters, for computational reasons we have limited our evaluation to only the feature subset that provided us with the highest MI, i.e., FS41-3.

| Feature Set | Useful Features | *SC* X 10⁷ | *vCV* X 10⁷ | *MCCV* X 10⁷ | *MI* |
|---|---|---|---|---|---|
| FS41-3 | 6,157 | **2.66** | 0.61 | 1.32 | **0.61** |
| FS52 | 386 | 2.8 | **0.3** | **0.69** | 0.51 |
| NF | 32,450 | 2.96 | 1.25 | 2.8 | 0.58 |
| LSI | 32450/165 | NA | NA | NA | 0.57 |

**Table 1 Comparison Of Results**

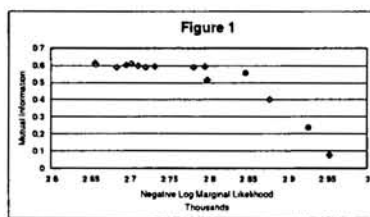

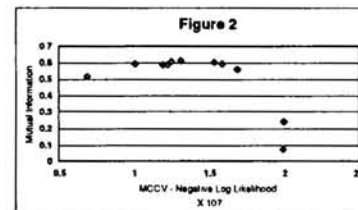

### 5.3 Discussion

The consistency between the MI and *SC* (Figure 1) is striking. The monotonic trend is more apparent at higher *SC* indicating that bad clusterings are more easily detected by *SC* while as the solution improves the differences are more subtle. *Note that the best value of SC and MI coincide.* Given the assumptions made in deriving equation (5), this consistency and is encouraging. The interested reader is referred to [18] for more details. Figures 2 and 3 indicate that there is certainly a reasonable consistency between the cross-validated likelihood and the MI although not as striking as the *SC*. Note that the MI for the feature sets picked by *MCCV* and *vCV* is significantly lower than that of the best feature-set. Figures 4,5 and 6 show the plots of SC, MCCV and vCV as the number of clusters is increased. Using *SC* we see that FS41-3 reveals an optimal structure around 40 clusters. As with feature selection, both *MCCV* and *vCV* obtain models of lower complexity than *SC*. Both show an optimum of about 30 clusters. More experiments are required before we draw final conclusions, however, the full Bayesian approach seems a practical and useful approach for model selection in document clustering. Our choice of likelihood function and priors provide a closed-form solution that is computationally tractable and provides meaningful results.

## 6.0 Conclusions

In this paper we tackled the problem of model structure determination in clustering. The main contribution of the paper is a Bayesian objective function that treats optimal model selection as choosing both the number of clusters *and* the feature subset. An important aspect of our work is a formal notion that forms a basis for doing feature selection in unsupervised learning. We then evaluated two approaches for model selection: one using this objective function and the other based on cross-validation.

Both approaches performed reasonably well - with the Bayesian scheme outperforming the cross-validation approaches in feature selection. More experiments using different parameter settings for the cross-validation schemes and different priors for the Bayesian scheme should result in better understanding and therefore more powerful applications of these approaches.

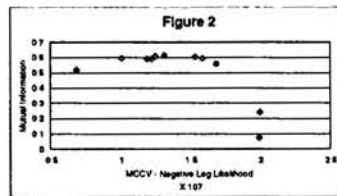

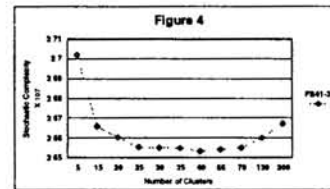

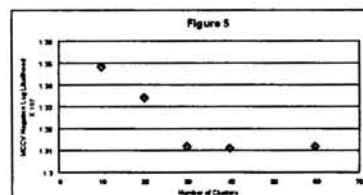

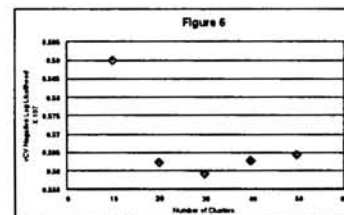

## References

[1] Baker, D., et al, Distributional Clustering of Words for Text Classification, SIGIR 1998.

[2] Bernardo, J. M. and Smith, A. F. M., Bayesian Theory, Wiley, 1994.

[3] Church, K.W. et al, Poisson Mixtures, Natural Language Engineering, I(12), 1995.

[4] Cover, T.M. and Thomas, J.A. Elements of Information Theory. Wiley-Interscience, 1991.

[5] Deerwester,S. et al, Indexing by Latent Semantic Analysis,JASIS, 1990.

[6] Dempster, A.et al., Maximum Likelihood from Incomplete Data Via the EM Algorithm, JRSS, 39,1977.

[7] Hanson,R., et al, Bayesian Classification with Correlation and Inheritance, IJCAI,1991.

[8] Iyengar, G., Clustering images using relative entropy for efficient retrieval, VLBV, 1998.

[9] Katz, S.M. , Distribution of content words and phrases in text and language modeling, NLE, 2, 1996.

[10] Kontkanen, P.T. et al, Comparing Bayesian Model Class Selection Criteria by Discrete Finite Mixtures, ISIS'96 Conference, 1996.

[11] Meila, M., Heckerman, D., An Experimental Comparison of Several Clustering and Initialization Methods, MSR-TR-98-06.

[12] Nigam, K et al, Learning to Classify Text from Labeled and Unlabeled Documents, AAAI, 1998.

[13] Pereira, F.C.N. et al, Distributional clustering of English words, ACL,1993.

[14] Rissanen, J., Stochastic Complexity in Statistical Inquiry. World\ Scientific, 1989.

[15] Rissanen, J., Ristad E., Unsupervised classification with stochastic complexity." The US/Japan Conference on the Frontiers of Statistical Modeling,1992.

[16] Singhal A. et al, Pivoted Document Length Normalization, SIGIR, 1996.

[17] Smyth, P., Clustering using Monte Carlo cross-validation, KDD, 1996.

[18] Vaithyanathan, S. and Dom, B. Model Selection in Unsupervised Learning with Applications to Document Clustering. IBM Research Report RJ-10137 (95012) Dec. 14, 1998 .